# Higher-Order Correlation Clustering for Image Segmentation

**Sungwoong Kim**
Department of EE, KAIST
Daejeon, South Korea
sungwoong.kim01@gmail.com

**Sebastian Nowozin**
Microsoft Research Cambridge
Cambridge, UK
Sebastian.Nowozin@microsoft.com

**Pushmeet Kohli**
Microsoft Research Cambridge
Cambridge, UK
pkohli@microsoft.com

**Chang D. Yoo**
Department of EE, KAIST
Daejeon, South Korea
cdyoo@ee.kaist.ac.kr

## Abstract

For many of the state-of-the-art computer vision algorithms, image segmentation is an important preprocessing step. As such, several image segmentation algorithms have been proposed, however, with certain reservation due to high computational load and many hand-tuning parameters. Correlation clustering, a graph-partitioning algorithm often used in natural language processing and document clustering, has the potential to perform better than previously proposed image segmentation algorithms. We improve the basic correlation clustering formulation by taking into account higher-order cluster relationships. This improves clustering in the presence of local boundary ambiguities. We first apply the pairwise correlation clustering to image segmentation over a pairwise superpixel graph and then develop higher-order correlation clustering over a hypergraph that considers higher-order relations among superpixels. Fast inference is possible by linear programming relaxation, and also effective parameter learning framework by structured support vector machine is possible. Experimental results on various datasets show that the proposed higher-order correlation clustering outperforms other state-of-the-art image segmentation algorithms.

## 1 Introduction

Image segmentation, a partitioning of an image into disjoint regions such that each region is homogeneous, is an important preprocessing step for many of the state-of-the-art algorithms for high-level image/scene understanding for three reasons. *First*, the coherent support of a region, commonly assumed to be of a single label, serves as a good prior for many labeling tasks. *Second*, these coherent regions allow a more consistent feature extraction that can incorporate surrounding contextual information by pooling many feature responses over the region. *Third*, compared to pixels, a small number of larger homogeneous regions significantly reduces the computational cost for a successive labeling task.

Image segmentation algorithms can be categorized into either non-graph-based or graph-based algorithms. Some well-known non-graph-based algorithms represented by mode-seeking algorithms such as the K-means [1], mean-shift [2], and EM [3] are available, while well-known graph-based algorithms are available as the min-cuts [4], normalized cuts [5] and Felzenszwalb-Huttenlocher (FH) segmentation algorithm [6]. In comparison to non-graph-based segmentations, *graph-based* segmentations have been shown to produce consistent segmentations by adaptively balancing local

judgements of similarity [7]. Moreover, the graph-based segmentation algorithms with *global objective* functions such as the min-cuts and normalized cuts have been shown to perform better than the FH algorithm that is based on the local objective function, since the global-objective algorithms benefit from the global nature of the information [7]. However, in contrast to the min-cuts and normalized cuts which are node-labeling algorithms, the FH algorithm benefits from the *edge-labeling* in that it leads to faster inference and does not require a pre-specified number of segmentations in each image [7].

Correlation clustering is a graph-partitioning algorithm [8] that simultaneously maximizes intra-cluster similarity and inter-cluster dissimilarity by solving the global objective (discriminant) function. In comparison with the previous image segmentation algorithms, correlation clustering is a graph-based, global-objective, and edge-labeling algorithm and therefore, has the potential to perform better for image segmentation. Furthermore, correlation clustering leads to the linear discriminant function which allows for approximate polynomial-time inference by linear programming (LP) and large margin training based on structured support vector machine (S-SVM) [9]. A framework that uses S-SVM for training the parameters in correlation clustering has been considered previously by Finley *et al.* [10]; however, the framework was applied to noun-phrase and news article clusterings. Taskar derived a max-margin formulation for learning the edge scores for correlation clustering [11]. However, his learning criterion is different from the S-SVM and is limited to applications involving two different segmentations of a single image. Furthermore, Taskar does not provide any experimental comparisons or quantitative results.

Even though the previous (pairwise) correlation clustering can consider global aspects of an image using the discriminatively-trained discriminant function, it is restricted in resolving the segment boundary ambiguities caused by neighboring pairwise relations presented by the pairwise graph. Therefore, to capture long-range dependencies of distant nodes in a global context, this paper proposes a novel *higher-order correlation clustering* to incorporate higher-order relations. We first apply the pairwise correlation clustering to image segmentation over a pairwise superpixel graph and then develop higher-order correlation clustering over a hypergraph that considers higher-order relations among superpixels.

The proposed higher-order correlation clustering is defined over a *hypergraph* in which an edge can connect to two or more nodes [12]. Hypergraphs have been previously used to lift certain limitations of conventional pairwise graphs [13, 14, 15]. However, previously proposed hypergraphs for image segmentation are restricted to partitioning based on the generalization of normalized cut framework, which suffer from a number of problems. *First*, inference is slow and difficult especially with increasing graph size. A number of algorithms to approximate the inference process have been introduced based on the coarsening algorithm [14] and the hypergraph Laplacian matrices [13]; these are heuristic approaches and therefore are sub-optimal. *Second*, incorporating a supervised learning algorithm for parameter estimation under the spectral hypergraph partitioning framework is difficult. This is in line with the difficulties in learning spectral graph partitioning. This requires a complex and unstable eigenvector approximation which must be differentiable [16, 17]. *Third*, utilizing rich region-based features is restricted. Almost all previous hypergraph-based image segmentation algorithms are restricted to use only color variances as region features.

The proposed higher-order correlation clustering overcomes all of these problems due to the generalization of the pairwise correlation clustering and enables to take advantages of using a hypergraph. The proposed higher-order correlation clustering algorithm uses as its input a hypergraph and leads to a linear discriminant function. A rich feature vector is defined based on several visual cues involving higher-order relations among superpixels. For fast inference, the LP relaxation is used to approximately solve the higher-order correlation clustering problem, and for supervised training of the parameter vector by S-SVM, we apply a decomposable structured loss function to handle unbalanced classes. We incorporate this loss function into the cutting plane procedure for S-SVM training. Experimental results on various datasets show that the proposed higher-order correlation clustering outperforms other state-of-the-art image segmentation algorithms.

The rest of the paper is organized as follows. Section 2 presents the higher-order correlation clustering for image segmentation. Section 3 describes large margin training for supervised image segmentation based on the S-SVM and the cutting plane algorithm. A number of experimental and comparative results are presented and discussed in Section 4, followed by a conclusion in Section 5.

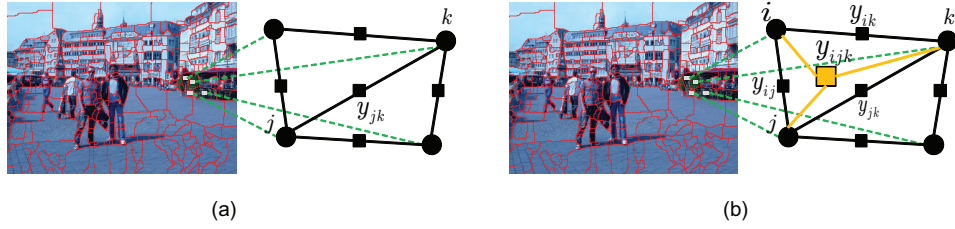

<div align="center">(a)                         (b)</div>

Figure 1: Illustrations of a part of (a) the pairwise graph (b) and the triplet graph built on superpixels.

## 2    Higher-order correlation clustering

The proposed image segmentation is based on superpixels which are small coherent regions preserving almost all boundaries between different regions, since superpixels significantly reduce computational cost and allow feature extraction to be conducted from a larger homogeneous region. The proposed correlation clustering merges superpixels into disjoint homogeneous regions over a superpixel graph.

### 2.1    Pairwise correlation clustering over pairwise superpixel graph

Define a pairwise undirected graph $\mathcal{G} = (\mathcal{V}, \mathcal{E})$ where a node corresponds to a superpixel and a link between adjacent superpixels corresponds to an edge (see Figure 1.(a)). A binary label $y_{jk}$ for an edge $(j, k) \in \mathcal{E}$ between nodes $j$ and $k$ is defined such that

$$y_{jk} = \begin{cases} 1, & \text{if nodes } j \text{ and } k \text{ belong to the same region,} \\ 0, & \text{otherwise.} \end{cases} \tag{1}$$

A discriminant function, which is the negative energy function, is defined over an image $\mathbf{x}$ and label $\mathbf{y}$ of all edges as

$$\begin{aligned} F(\mathbf{x}, \mathbf{y}; \mathbf{w}) &= \sum_{(j,k) \in \mathcal{E}} \text{Sim}_{\mathbf{w}}(\mathbf{x}, j, k) y_{jk} \\ &= \sum_{(j,k) \in \mathcal{E}} \langle \mathbf{w}, \phi_{jk}(\mathbf{x}) \rangle y_{jk} = \langle \mathbf{w}, \sum_{(j,k) \in \mathcal{E}} \phi_{jk}(\mathbf{x}) y_{jk} \rangle = \langle \mathbf{w}, \Phi(\mathbf{x}, \mathbf{y}) \rangle \end{aligned} \tag{2}$$

where the similarity measure between nodes $j$ and $k$, $\text{Sim}_{\mathbf{w}}(\mathbf{x}, j, k)$, is parameterized by $\mathbf{w}$ and takes values of both signs such that a large positive value means strong similarity while a large negative value means high degree of dissimilarity. Note that the discriminant function $F(\mathbf{x}, \mathbf{y}; \mathbf{w})$ is assumed to be linear in both the parameter vector $\mathbf{w}$ and the joint feature map $\Phi(\mathbf{x}, \mathbf{y})$, and $\phi_{jk}(\mathbf{x})$ is a pairwise feature vector which reflects the correspondence between the $j$th and the $k$th superpixels. An image segmentation is to infer the edge label, $\hat{\mathbf{y}}$, over the pairwise superpixel graph $\mathcal{G}$ by maximizing $F$ such that

$$\hat{\mathbf{y}} = \underset{\mathbf{y} \in \mathcal{Y}}{\text{argmax}} \, F(\mathbf{x}, \mathbf{y}; \mathbf{w}) \tag{3}$$

where $\mathcal{Y}$ is the set of $\{0, 1\}^{\mathcal{E}}$ that corresponds to a *valid segmentation*, the so called multicut polytope. However, solving (3) with this $\mathcal{Y}$ is generally NP-hard. Therefore, we approximate $\mathcal{Y}$ by means of a common multicut LP relaxation [18] with the following two constraints: (1) cycle inequality and (2) odd-wheel inequality. When producing the segmentation results based on the approximated LP solutions, we take the floor of a fractionally-predicted label of each edge independently for simply obtaining valid integer solutions that may be sub-optimal.

Even though this pairwise correlation clustering takes a rich pairwise feature vector and a trained parameter vector (which will be presented later), it often produces incorrectly predicted segments due to the segment boundary ambiguities caused by limited pairwise relations of neighboring superpixels (see Figure 2). Therefore, to incorporate higher-order relations, we develop higher-order correlation clustering by generalizing the correlation clustering over a hypergraph.

### 2.2    Higher-order correlation clustering over hypergraph

The proposed higher-order correlation clustering is defined over a hypergraph in which an edge called *hyperedge* can connect to two or more nodes. For example, as shown in Figure 1.(b), one

<div align="center">3</div>

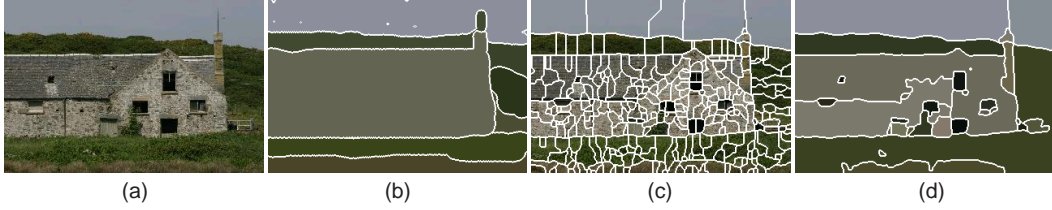

| (a) | (b) | (c) | (d) |

Figure 2: Example of segmentation result by pairwise correlation clustering. (a) Original image. (b) Ground-truth. (c) Superpixels. (d) Segments obtained by pairwise correlation clustering.

can introduce binary labels for each adjacent vertices forming a triplet such that $y_{ijk} = 1$ if all vertices in the triplet ($\{i, j, k\}$) are in the same cluster; otherwise, $y_{ijk} = 0$. Define a hypergraph $\mathcal{HG} = (\mathcal{V}, \mathcal{E})$ where $\mathcal{V}$ is a set of nodes (superpixels) and $\mathcal{E}$ is a set of hyperedges (subsets of $\mathcal{V}$) such that $\bigcup_{e \in \mathcal{E}} = \mathcal{V}$. Here, a hyperedge $e$ has at least two nodes, i.e. $|e| \geq 2$. Therefore, the hyperedge set $\mathcal{E}$ can be divided into two disjoint subsets: pairwise edge set $\mathcal{E}_p = \{e \in \mathcal{E} \mid |e| = 2\}$ and higher-order edge set $\mathcal{E}_h = \{e \in \mathcal{E} \mid |e| > 2\}$ such that $\mathcal{E}_p \bigcup \mathcal{E}_h = \mathcal{E}$. Note that in the proposed hypergraph for higher-order correlation clustering all hyperedges containing just two nodes ($\forall e_p \in \mathcal{E}_p$) are linked between adjacent superpixels. The pairwise superpixel graph is a special hypergraph where all hyperedges contain just two (neighboring) superpixels: $\mathcal{E}_p = \mathcal{E}$. A binary label $y_e$ for a hyperedge $e \in \mathcal{E}$ is defined such that

$$y_e = \begin{cases} 1, & \text{if all nodes in } e \text{ belong to the same region,} \\ 0, & \text{otherwise.} \end{cases} \tag{4}$$

Similar to the pairwise correlation clustering, a linear discriminant function is defined over an image $\mathbf{x}$ and label $\mathbf{y}$ of all hyperedges as

$$
\begin{aligned}
F(\mathbf{x}, \mathbf{y}; \mathbf{w}) &= \sum_{e \in \mathcal{E}} \mathrm{Hom}_{\mathbf{w}}(\mathbf{x}, e) y_e \\
&= \sum_{e \in \mathcal{E}} \langle \mathbf{w}, \phi_e(\mathbf{x}) \rangle y_e = \sum_{e_p \in \mathcal{E}_p} \langle \mathbf{w}_p, \phi_{e_p}(\mathbf{x}) \rangle y_{e_p} + \sum_{e_h \in \mathcal{E}_h} \langle \mathbf{w}_h, \phi_{e_h}(\mathbf{x}) \rangle y_{e_h} = \langle \mathbf{w}, \Phi(\mathbf{x}, \mathbf{y}) \rangle \quad (5)
\end{aligned}
$$

where the homogeneity measure among nodes in $e$, $\mathrm{Hom}_{\mathbf{w}}(\mathbf{x}, e)$, is also the inner product of the parameter vector $\mathbf{w}$ and the feature vector $\phi_e(\mathbf{x})$ and takes values of both signs such that a large positive value means strong homogeneity while a large negative value means high degree of non-homogeneity. Note that the proposed discriminant function for higher-order correlation clustering is decomposed into two terms by assigning different parameter vectors to the pairwise edge set $\mathcal{E}_p$ and the higher-order edge set $\mathcal{E}_h$ such that $\mathbf{w} = [\mathbf{w}_p^T, \mathbf{w}_h^T]^T$. Thus, in addition to the pairwise similarity between neighboring superpixels, the proposed higher-order correlation clustering considers a broad homogeneous region reflecting higher-order relations among superpixels.

Now the problem is how to build our hypergraph from a given image. Here, we use unsupervised multiple partitionings (quantizations) from baseline superpixels. We obtain unsupervised multiple partitionings by merging not pixels but superpixels with different image quantizations using the ultrametric contour maps [19]. For example, in Figure 3, there are three region layers, one superpixel (pairwise) layer and two higher-order layers, from which a hypergraph is constructed by defining hyperedges as follows: first, all edges (black line) in the pairwise superpixel graph from the first layer are incorporated into the pairwise edge set $\mathcal{E}_p$, while hyperedges (yellow line) corresponding to regions (groups of superpixels) in the second and third layers are included in the higher-order edge set $\mathcal{E}_h$. Note that we can further decompose the higher-order term in (5) into two terms associated with the second layer and the third layer, respectively, by assigning different parameter vectors; however for simplicity, this paper aggregates all higher-order edges from all higher-order layers into a single higher-order edge set assigning the same parameter vector.

### 2.2.1  LP relaxation for inference

An image segmentation is to infer the hyperedge label, $\hat{\mathbf{y}}$, over the hypergraph $\mathcal{HG}$ by maximizing the discriminant function $F$ such that

$$\hat{\mathbf{y}} = \operatorname*{argmax}_{\mathbf{y} \in \mathcal{Y}} F(\mathbf{x}, \mathbf{y}; \mathbf{w}) \tag{6}$$

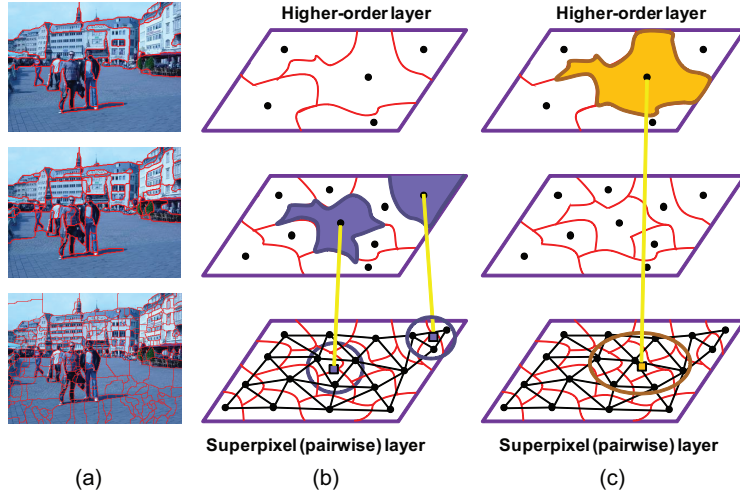

Higher-order layer      Higher-order layer

Superpixel (pairwise) layer      Superpixel (pairwise) layer

(a)      (b)      (c)

Figure 3: Hypergraph construction from multiple partitionings. (a) Multiple partitionings from baseline superpixels. (b) Hyperedge (yellow line) corresponding to a region in the second layer. (c) Hyperedge (yellow line) corresponding to a region in the third layer.

where $\mathcal{Y}$ is also the set of $\{0,1\}^{\mathcal{E}}$ that corresponds to a *valid segmentation*. Since the inference problem (6) is also NP-hard, we relax $\mathcal{Y}$ by (facet-defining) linear inequalities. In addition to the constraints placed on pairwise labels such that the cycle inequality and odd-wheel inequality hold pairwise correlation clustering, we augment the constraints for labels on the higher-order edges, called *higher-order inequalities*, for a valid segmentation; there is no all-one pairwise labels in a region for which the higher-order edge is labeled as zero (non-homogeneous region), and when a region is labeled as one (homogeneous region), all pairwise labels in that region should be one. These higher-order inequalities can be formulated as

$$y_{e_h} \leq y_{e_p}, \;\; \forall e_p \in \mathcal{E}_p | e_p \subset e_h, \tag{7}$$

$$(1 - y_{e_h}) \leq \sum_{e_p \in \mathcal{E}_p | e_p \subset e_h} (1 - y_{e_p}).$$

Indeed, the LP relaxation to approximately solve (6) is formulated as

$$\underset{\mathbf{y}}{\operatorname{argmax}} \sum_{e_p \in \mathcal{E}_p} \langle \mathbf{w}_p, \phi_{e_p}(\mathbf{x}) \rangle y_{e_p} + \sum_{e_h \in \mathcal{E}_h} \langle \mathbf{w}_h, \phi_{e_h}(\mathbf{x}) \rangle y_{e_h} \tag{8}$$

$$\text{s.t.} \quad \forall\, e \in \mathcal{E}(=\mathcal{E}_p \bigcup \mathcal{E}_h), \;\; y_e \in [0,1],$$

$$\forall\, e_p \in \mathcal{E}_p, \;\; \text{cycle inequalities, odd-wheel inequalities [18]},$$

$$\forall\, e_h \in \mathcal{E}_h, \;\; \text{higher-order inequalities (7)}.$$

Note that the proposed higher-order correlation clustering follows the concept of *soft constraints*: superpixels within a hyperedge are encouraged to merge if a hyperedge is highly homogeneous.

### 2.2.2   Feature vector

We construct a 771-dimensional feature vector $\phi_e(\mathbf{x})$ by concatenating several visual cues with different quantization levels and thresholds. The pairwise feature vector $\phi_{e_p}(\mathbf{x})$ reflects the correspondence between neighboring superpixels, and the higher-order feature vector $\phi_{e_h}(\mathbf{x})$ characterizes a more complex relations among superpixels in a broader region to measure homogeneity. The magnitude of $\mathbf{w}$ determines the importance of each feature, and this importance is task-dependent. Thus, $\mathbf{w}$ is estimated by supervised training described in Section 3.

1. Pairwise feature vector (611-dim): $\phi_{e_p} = [\phi_{e_p}^c; \phi_{e_p}^t; \phi_{e_p}^s; \phi_{e_p}^e; \phi_{e_p}^v; 1]$.
   - Color difference $\phi^c$: The 26 RGB/HSV color distances (absolute differences, $\chi^2$-distances, earth mover's distances) between two adjacent superpixels.

- Texture difference $\phi^t$: The 64 texture distances (absolute differences, $\chi^2$-distances, earth mover's distances) between two adjacent superpixels using 15 Leung-Malik (LM) filter banks [19].
- Shape/location difference $\phi^s$: The 5-dimensional shape/location feature proposed in [20].
- Edge strength $\phi^e$: The 1-of-15 coding of the quantized edge strength proposed in [19].
- Joint visual word posterior $\phi^v$: The 100-dimensional vector holding the joint visual word posteriors for a pair of neighboring superpixels using 10 visual words and the 400-dimensional vector holding the joint posteriors based on 20 visual words [21].

2. Higher-order feature vector (160-dim): $\phi_{e_h} = [\phi_{e_h}^{va}; \phi_{e_h}^{e}; \phi_{e_h}^{tm}; 1]$.

- Variance $\phi^{va}$: The 14 color variances and 30 texture variances among superpixels in a hyperedge.
- Edge strength $\phi^e$: The 1-of-15 coding of the quantized edge strength proposed in [19].
- Template matching score $\phi^{tm}$: The color/texture and shape/location features of all regions in the training images are clustered using $k$-means with $k = 100$ to obtain 100 representative templates of distinct regions. The 100-dimensional template matching feature vector is composed of the matching scores between a region defined by a hyperedge and templates using the Gaussian RBF kernel.

In each feature vector, the bias (=1) is augmented for proper similarity/homogeneity measure which can either be positive or negative.

## 3 Structural learning

The proposed discriminant function is defined over the superpixel graph, and therefore, the ground-truth segmentation needs to be transformed to the ground-truth edge labels in the superpixel graph. For this, we first assign a single dominant segment label to each superpixel by majority voting over the superpixel's constituent pixels and then obtain the ground-truth edge labels.

Using this ground-truth edge labels of the training data, the S-SVM [9] is performed to estimate the parameter vector. Given $N$ training samples $\{\mathbf{x}^n, \mathbf{y}^n\}_{n=1}^N$ where $\mathbf{y}^n$ is the ground-truth edge labels for the $n$th training image, the S-SVM [9] optimizes $\mathbf{w}$ by minimizing a quadratic objective function subject to a set of linear margin constraints:

$$\min_{\mathbf{w}, \xi} \quad \frac{1}{2}\|\mathbf{w}\|^2 + C \sum_{n=1}^N \xi_n \tag{9}$$
$$\text{s.t.} \quad \langle \mathbf{w}, \Delta\Phi(\mathbf{x}^n, \mathbf{y}) \rangle \geq \Delta(\mathbf{y}^n, \mathbf{y}) - \xi_n, \ \forall n, \mathbf{y} \in \mathcal{Y}\backslash\mathbf{y}^n,$$
$$\xi_n \geq 0, \quad \forall n$$

where $\Delta\Phi(\mathbf{x}^n, \mathbf{y}) = \Phi(\mathbf{x}^n, \mathbf{y}^n) - \Phi(\mathbf{x}^n, \mathbf{y})$, and $C > 0$ is a constant that controls the trade-off between margin maximization and training error minimization. In the S-SVM, the margin is scaled with a loss $\Delta(\mathbf{y}^n, \mathbf{y})$, which is the difference measure between prediction $\mathbf{y}$ and ground-truth label $\mathbf{y}^n$ of the $n$th image. The S-SVM offers good generalization ability as well as the flexibility to choose any loss function [9].

The cutting plane algorithm [9, 18] with LP relaxation for loss-augmented inference is used to solve the optimization problem of S-SVM, since fast convergence and high robustness of the cutting plane algorithm in handling a large number of margin constraints are well-known [22, 23].

A loss function is usually a non-negative function, and a loss function that is decomposable is preferred, since it enables the loss-augmented inference in the cutting plane algorithm to be performed efficiently. The most popular loss function that is decomposable is the Hamming distance which is equivalent to the number of mismatches between $\mathbf{y}^n$ and $\mathbf{y}$ at the edge level in this correlation clustering. Unfortunately, in the proposed correlation clustering for image segmentation, the number of edges which are labeled as 1 is considerably higher than that of edges which are labeled as 0. This unbalance makes other learning methods such as the perceptron algorithm inappropriate, since it leads to the clustering of the whole image as one segment. This problem due to the unbalance also

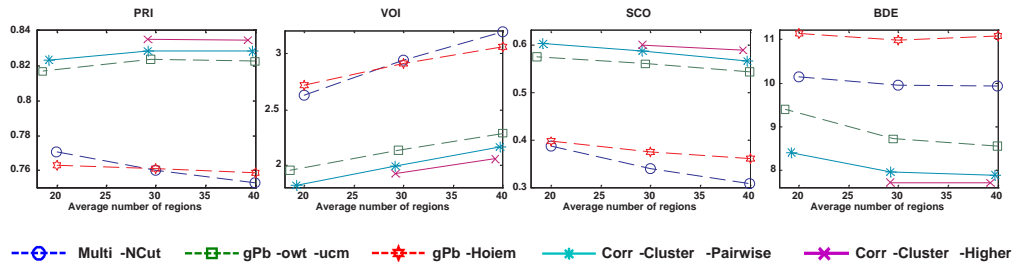

Figure 4: Obtained evaluation measures from segmentation results on the SBD.

occurs when we use the Hamming loss in the S-SVM. Therefore, we use the following loss function:

$$\Delta(\mathbf{y}^n, \mathbf{y}) = \sum_{e_p \in \mathcal{E}_p} \Big( R_p\, y_{e_p}^n + y_{e_p} - (R_p + 1) y_{e_p}^n y_{e_p} \Big) + D \sum_{e_h \in \mathcal{E}_h} \Big( R_h\, y_{e_h}^n + y_{e_h} - (R_h + 1) y_{e_h}^n y_{e_h} \Big) \quad (10)$$

where $D$ is the relative weight of the loss at higher-order edge level to that of the loss at pairwise edge level. In addition, $R_p$ and $R_h$ control the relative importance between the incorrect merging of the superpixels and the incorrect separation of the superpixels by imposing different weights to the false negative and the false positive. Here, we set both $R_p$ and $R_h$ to be less than 1 to overcome the problem due to the unbalance.

## 4   Experiments

To evaluate segmentations obtained by various algorithms against the ground-truth segmentation, we conducted image segmentations on three benchmark datasets: Stanford background dataset [24] (SBD), Berkeley segmentation dataset (BSDS) [25], MSRC dataset [26]. For image segmentation based on correlation clustering, we initially obtain baseline superpixels (438 superpixels per image on average) by the gPb contour detector and the oriented watershed transform [19] and then construct a hypergraph. The function parameters are initially set to zero, and then based on the S-SVM, the structured output learning is used to estimate the parameter vectors. Note that the relaxed solutions in loss-augmented inference are used during training, while in testing, our simple rounding method is used to produce valid segmentation results. Rounding is only necessary in case we obtain fractional solutions from LP-relaxed correlation clustering.

We compared the proposed pairwise/higher-order correlation clustering to the following state-of-the-art image segmentation algorithms: multiscale NCut [27], gPb-owt-ucm [19], and gPb-Hoiem [20] that grouped the same superpixels based on pairwise same-label likelihoods. The pairwise same-label likelihoods were independently learnt from the training data with the same 611-dimensional pairwise feature vector. We consider four performance measures: *probabilistic Rand index* (PRI) [28], *variation of information* (VOI) [29], *segmentation covering* (SCO) [19], and *boundary displacement error* (BDE) [30]. As the predicted segmentation is close to the ground-truth segmentation, the PRI and SCO are increased while the VOI and BDE are decreased.

### 4.1   Stanford background dataset

The SBD consists of 715 outdoor images with corresponding pixel-wise annotations. We employed 5-fold cross-validation with the dataset randomly split into 572 training images and 143 test images for each fold. Figure 4 shows the obtained four measures from segmentation results according to the average number of regions. Note that the performance varies with different numbers of regions, and for this reason, we designed each algorithm to produce multiple segmentations (20 to 40 regions). Specifically, multiple segmentations in the proposed algorithm were obtained by varying $R_p$ (0.001~0.2) and $R_h$ (0.1~1.0) in the loss function during training ($D$=10). Irrespective of the measure, the proposed higher-order correlation clustering (Corr-Cluster-Higher) performed better than other algorithms including the pairwise correlation clustering (Corr-Cluster-Pairwise). Figure 5 shows some example segmentations. The proposed higher-order correlation clustering yielded the best segmentation results. In specific, incorrectly predicted segments by pairwise correlation clustering were reduced in the segmentation results obtained by higher-order correlation clustering

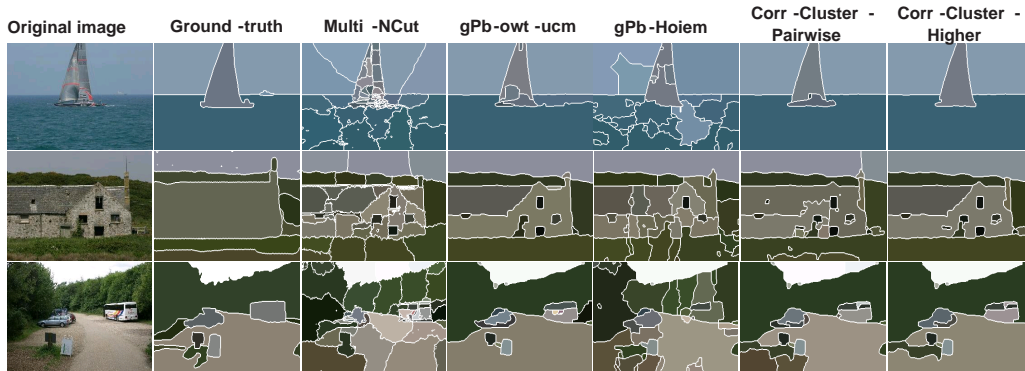

Figure 5: Results of image segmentation.

Table 1: Quantitative results on the BSDS test set and on the MSRC test set.

|  | BSDS | | | | MSRC | | | |
| --- | --- | --- | --- | --- | --- | --- | --- | --- |
| Test set | PRI | VOI | SCO | BDE | PRI | VOI | SCO | BDE |
| Multi-NCut | 0.728 | 3.043 | 0.315 | 14.257 | 0.628 | 2.765 | 0.341 | 11.941 |
| gPb-owt-ucm | 0.794 | 1.909 | 0.571 | 11.461 | 0.779 | 1.675 | 0.628 | 9.800 |
| gPb-Hoiem | 0.724 | 3.194 | 0.316 | 14.795 | 0.614 | 2.847 | 0.353 | 13.533 |
| Corr-Cluster-Pairwise | 0.806 | 1.829 | 0.585 | 11.194 | 0.773 | 1.648 | 0.632 | 9.194 |
| Corr-Cluster-Higher | **0.814** | **1.743** | **0.599** | **10.377** | **0.784** | **1.594** | **0.648** | **9.040** |

owing to the consideration of higher-order relations in broad regions. Regarding the runtime of our algorithm, we observed that for test-time inference it took on average around 15 seconds (graph construction and feature extraction: 14s, LP: 1s) per image on a 2.67GHz processor, whereas the overall training took 10 hours on the training set. Note that other segmentation algorithms such as the multiscale NCut and the gPb-owt-ucm took on average a few minutes per image.

## 4.2 Berkeley segmentation dataset and MSRC dataset

The BSDS contains 300 natural images split into the 200 training images and 100 test images. Since each image is segmented by multiple human subjects, we defined a single probabilistic (real-valued) ground-truth segmentation of each image for training in the proposed correlation clustering. The MSRC dataset is composed of 591 natural images. We split the data into 45% training, 10% validation, and 45% test sets, following [26]. The performance was evaluated using the clean ground-truth object instance labeling of [31]. On average, all segmentation algorithms were set to produce 30 disjoint regions per image on the BSDS and 15 disjoint regions per image on the MSRC dataset. As shown in Table 1, the proposed higher-order correlation clustering gave the best results on both datasets. Especially, the obtained results on the BSDS are similar or even better than the best results ever reported on the BSDS [32, 19].

## 5 Conclusion

This paper proposed the higher-order correlation clustering over a hypergraph to merge superpixels into homogeneous regions. The LP relaxation was used to approximately solve the higher-order correlation clustering over a hypergraph where a rich feature vector was defined based on several visual cues involving higher-order relations among superpixels. The S-SVM was used for supervised training of parameters in correlation clustering, and the cutting plane algorithm with LP-relaxed inference was applied to solve the optimization problem of S-SVM. Experimental results showed that the proposed higher-order correlation clustering outperformed other image segmentation algorithms on various datasets. The proposed framework is applicable to a variety of other areas.

### Acknowledgments

This work was supported by the National Research Foundation of Korea (NRF) grant funded by the Korea government (MEST) (No.2011-0018249).

# References

[1] T. Kanungo, D. Mount, N. Netanyahu, C. Piatko, R. Silverman, and A. Wu, "An efficient k-means clustering algorithm: Analysis and implementation," *PAMI*, vol. 24, pp. 881–892, 2002.

[2] D. Comaniciu and P. Meer, "Mean shift: A robust approach toward feature space analysis," *PAMI*, vol. 24, pp. 603–619, 2002.

[3] C. Carson, S. Belongie, H. Greenspan, and J. Malik, "Blobworld: image segmentation using expectation-maximization and its application to image querying," *PAMI*, vol. 24, pp. 1026–1038, 2002.

[4] F. Estrada and A. Jepson, "Spectral embedding and mincut for image segmentation," in *BMVC*, 2004.

[5] J. Shi and J. Malik, "Normalized cuts and image segmentation," *PAMI*, vol. 22, pp. 888–905, 2000.

[6] P. Felzenszwalb and D. Huttenlocher, "Efficient graph-based image segmentation," *IJCV*, vol. 59, pp. 167–181, 2004.

[7] F. Estrada and A. Jepson, "Benchmarking image segmentation algorithms," *IJCV*, vol. 85, 2009.

[8] N. Bansal, A. Blum, and S. Chawla, "Correlation clustering," *Machine Learning*, vol. 56, 2004.

[9] I. Tsochantaridis, T. Joachims, T. Hofmann, and Y. Altun, "Large margin methods for structured and independent output variables," *JMLR*, vol. 6, 2005.

[10] T. Finley and T. Joachims, "Supervised clustering with support vector machines," in *ICML*, 2005.

[11] B. Taskar, "Learning structured prediction models: a large margin approach," *Ph.D. thesis, Stanford University*, 2004.

[12] C. Berge, *Hypergraphs*, North-Holland, Amsterdam, 1989.

[13] L. Ding and A. Yilmaz, "Image segmentation as learning on hypergraphs," in *Proc. ICMAL*, 2008.

[14] S. Rital, "Hypergraph cuts and unsupervised representation for image segmentation," *Fundamenta Informaticae*, vol. 96, pp. 153–179, 2009.

[15] A. Ducournau, S. Rital, A. Bretto, and B. Laget, "A multilevel spectral hypergraph partitioning approach for color image segmentation," in *Proc. ICSIPA*, 2009.

[16] F. Bach and M. I. Jordan, "Learning spectral clustering," in *NIPS*, 2003.

[17] T. Cour, N. Gogin, and J. Shi, "Learning spectral graph segmentation," in *AISTATS*, 2005.

[18] S. Nowozin and S. Jegelka, "Solution stability in linear programming relaxations: Graph partitioning and unsupervised learning," in *ICML*, 2009.

[19] P. Arbeláez, M. Maire, C. Fowlkes, and J. Malik, "Contour detection and hierarchical image segmentation," *PAMI*, vol. 33, pp. 898–916, 2011.

[20] D. Hoiem, A. A. Efros, and M. Hebert, "Recovering surface layout from an image," *IJCV*, 2007.

[21] D. Batra, R. Sukthankar, and T. Chen, "Learning class-specific affinities for image labelling," in *CVPR*, 2008.

[22] T. Finley and T. Joachims, "Training structural SVMs when exact inference is intractable," in *ICML*, 2008.

[23] A. Kulesza and F. Pereira, "Structured learning with approximate inference," in *NIPS*, 2007.

[24] S. Gould, R. Fulton, and D. Koller, "Decomposing a scene into geometric and semantically consistent regions," in *ICCV*, 2009.

[25] C. Fowlkes, D. Martin, and J. Malik, *The Berkeley Segmentation Dataset and Benchmark (BSDB)*, http://www.cs.berkeley.edu/projects/vision/grouping/segbench/.

[26] J. Shotton, J. Winn, C. Rother, and A. Criminisi, "Textonboost: joint apprearence, shape and context modeling for multi-class object recognition and segmentation," in *ECCV*, 2006.

[27] T. Cour, F. Benezit, and J. Shi, "Spectral segmentation with multiscale graph decomposition," in *CVPR*, 2005.

[28] W. M. Rand, "Objective criteria for the evaluation of clustering methods," *Journal of the American Statistical Association*, vol. 66, pp. 846–850, 1971.

[29] M. Meila, "Computing clusterings: An axiomatic view," in *ICML*, 2005.

[30] J. Freixenet, X. Munoz, D. Raba, J. Marti, and X. Cufi, "Yet another survey on image segmentation: Region and boundary information integration," in *ECCV*, 2002.

[31] T. Malisiewicz and A. A. Efros, "Improving spatial support for objects via multiple segmentations," in *BMVC*, 2007.

[32] T. Kim, K. Lee, and S. Lee, "Learning full pairwise affinities for spectral segmentation," in *CVPR*, 2010.

